# Optimal Bayesian Recommendation Sets and Myopically Optimal Choice Query Sets

**Paolo Viappiani**[*]
Department of Computer Science
University of Toronto
paolo.viappiani@gmail.com

**Craig Boutilier**
Department of Computer Science
University of Toronto
cebly@cs.toronto.edu

## Abstract

Bayesian approaches to utility elicitation typically adopt (myopic) expected value of information (EVOI) as a natural criterion for selecting queries. However, EVOI-optimization is usually computationally prohibitive. In this paper, we examine EVOI optimization using *choice queries*, queries in which a user is ask to select her most preferred product from a set. We show that, under very general assumptions, the optimal choice query w.r.t. EVOI coincides with the *optimal recommendation set*, that is, a set maximizing the expected utility of the user selection. Since recommendation set optimization is a simpler, submodular problem, this can greatly reduce the complexity of both exact and approximate (greedy) computation of optimal choice queries. We also examine the case where user responses to choice queries are error-prone (using both constant and mixed multinomial logit noise models) and provide worst-case guarantees. Finally we present a local search technique for query optimization that works extremely well with large outcome spaces.

## 1 Introduction

Utility elicitation is a key component in many decision support applications and recommender systems, since appropriate decisions or recommendations depend critically on the preferences of the user on whose behalf decisions are being made. Since full elicitation of user utility is prohibitively expensive in most cases (w.r.t. time, cognitive effort, etc.), we must often rely on partial utility information. Thus in *interactive preference elicitation*, one must selectively decide which queries are most informative relative to the goal of making good or optimal recommendations. A variety of principled approaches have been proposed for this problem. A number of these focus directly on (myopically or heuristically) reducing uncertainty regarding utility parameters as quickly as possible, including max-margin [10], volumetric [12], polyhedral [22] and entropy-based [1] methods.

A different class of approaches does not attempt to reduce utility uncertainty for its own sake, but rather focuses on discovering utility information that improves the quality of the recommendation. These include regret-based [3, 23] and Bayesian [7, 6, 2, 11] models. We focus on Bayesian models in this work, assuming some prior distribution over user utility parameters and conditioning this distribution on information acquired from the user (e.g., query responses or behavioral observations). The most natural criterion for choosing queries is *expected value of information (EVOI)*, which can be optimized myopically [7] or sequentially [2]. However, optimization of EVOI for online query selection is not feasible except in the most simple cases. Hence, in practice, heuristics are used that offer no theoretical guarantees with respect to query quality.

In this paper we consider the problem of myopic EVOI optimization using *choice queries*. Such queries are commonly used in conjoint analysis and product design [15], requiring a user to indicate which choice/product is most preferred from a set of $k$ options. We show that, under very general assumptions, optimization of choice queries reduces to the simpler problem of choosing the *optimal recommendation set*, i.e., the set of $k$ products such that, if a user were forced to choose one,

---

[*]From 9/2010 to 12/2010 at the University of Regina; from 01/2011 onwards at Aalborg University.

maximizes utility of that choice (in expectation). Not only is the optimal recommendation set problem somewhat easier computationally, it is submodular, admitting a greedy algorithm with approximation guarantees. Thus, it can be used to determine approximately optimal choice queries. We develop this connection under several different (noisy) user response models. Finally, we describe *query iteration*, a local search technique that, though it has no formal guarantees, finds near-optimal recommendation sets and queries much faster than either exact or greedy optimization.

## 2 Background: Bayesian Recommendation and Elicitation

We assume a system is charged with the task of recommending an option to a user in some multi-attribute space, for instance, the space of possible product configurations from some domain (e.g., computers, cars, rental apartments, etc.). Products are characterized by a finite set of attributes $\mathcal{X} = \{X_1, ...X_n\}$, each with finite domain $Dom(X_i)$. Let $\mathbf{X} \subseteq Dom(\mathcal{X})$ denote the set of *feasible configurations*. For instance, attributes may correspond to the features of various cars, such as color, engine size, fuel economy, etc., with $\mathbf{X}$ defined either by constraints on attribute combinations (e.g., constraints on computer components that can be put together) or by an explicit database of feasible configurations (e.g., a rental database). The user has a *utility function* $u : Dom(\mathcal{X}) \rightarrow \mathbf{R}$. The precise form of $u$ is not critical, but in our experiments we assume that $u(\mathbf{x}; w)$ is linear in the parameters (or weights) $w$ (e.g., as in generalized additive independent (GAI) models [8, 5].) We often refer to $w$ as the user's "utility function" for simplicity, assuming a fixed form for $u$. A simple additive model in the car domain might be:

$$u(Car; w) = w_1 f_1(MPG) + w_2 f_2(EngineSize) + w_3 f_3(Color).$$

The optimal product $x_w^*$ for a user with utility parameters $w$ is the $x \in \mathbf{X}$ that maximizes $u(x; w)$.

Generally, a user's utility function $w$ will not be known with certainty. Following recent models of Bayesian elicitation, the system's uncertainty is reflected in a distribution, or *beliefs*, $P(w; \theta)$ over the space $W$ of possible utility functions [7, 6, 2]. Here $\theta$ denotes the parameterization of our model, and we often refer to $\theta$ as our *belief state*. Given $P(\cdot; \theta)$, we define the *expected utility* of an option $x$ to be $EU(x; \theta) = \int_W u(x; w) P(w; \theta) dw$. If required to make a recommendation given belief $\theta$, the optimal option $x^*(\theta)$ is that with greatest expected utility $EU^*(\theta) = \max_{x \in X} EU(x; \theta)$, with $x^*(\theta) = \arg\max_{x \in X} EU(x; \theta)$.

In some settings, we are able to make *set-based recommendations*: rather than recommending a single option, a small set of $k$ options can be presented, from which the user selects her most preferred option [15, 20, 23]. We discuss the problem of constructing an optimal *recommendation set* $S$ further below. Given recommendation set $S$ with $x \in S$, let $S \triangleright x$ denote that $x$ has the greatest utility among those items in $S$ (for a given utility function $w$). Given feasible utility space $W$, we define $W \cap S \triangleright x \equiv \{w \in W : u(x; w) \geq u(y; w), \forall y \neq x, y \in S\}$ to be those utility functions satisfying $S \triangleright x$. Ignoring "ties" over full-dimensional subsets of $W$ (which are easily dealt with, but complicate the presentation), the regions $W \cap S \triangleright x_i, x_i \in S$, partition utility space.

A recommender system can refine its belief state $\theta$ by learning more about the user's utility function $w$. A reduction in uncertainty will lead to better recommendations (in expectation). While many sources of information can be used to assess a user preferences—including the preferences of related users, as in collaborative filtering [14], or observed user choice behavior [15, 19]—we focus on explicit *utility elicitation*, in which a user is asked questions about her preferences.

There are a variety of query types that can be used to refine one's knowledge of a user's utility function (we refer to [13, 3, 5] for further discussion). *Comparison queries* are especially natural, asking a user if she prefers one option $x$ to another $y$. These comparisons can be localized to specific (subsets of) attributes in additive or GAI models, and such structured models allow responses w.r.t. specific options to "generalize," providing constraints on the utility of related options. In this work we consider the extension of comparisons to *choice sets* of more than two options [23] as is common in conjoint analysis [15, 22]. Any set $S$ can be interpreted as a query: the user states which of the $k$ elements $x_i \in S$ she prefers. We refer to $S$ interchangeably as a *query* or a *choice set*.

The user's response to a choice set tells us something about her preferences; but this depends on the *user response model*. In a *noiseless model*, the user correctly identifies the preferred item in the slate: the choice of $x_i \in S$ refines the set of feasible utility functions $W$ by imposing $k - 1$ linear constraints of the form $u(x_i; w) \geq u(x_j; w), j \neq i$, and the new belief state is obtained by

restricting $\theta$ to have non-zero density only on $W \cap S \triangleright x_i$ and renormalizing. More generally, a *noisy response model* allows that a user may select an option that does not maximize her utility. For any choice set $S$ with $x_i \in S$, let $S \rightsquigarrow x_i$ denote the event of the user selecting $x_i$. A response model $R$ dictates, for any choice set $S$, the probability $P_R(S \rightsquigarrow x_i; w)$ of any selection given utility function $w$. When the beliefs about a user's utility are uncertain, we define $P_R(S \rightsquigarrow x_i; \theta) = \int_W P_R(S \rightsquigarrow x_i; w)P(w; \theta)dw$. We discuss various response models below.

When treating $S$ as a query set (as opposed to a recommendation set), we are not interested in its expected utility, but rather in its *expected value of information (EVOI)*, or the (expected) degree to which a response will increase the quality of the system's recommendation. We define:

**Definition 1** *Given belief state $\theta$, the* expected posterior utility $(EPU)$ *of query set $S$ under $R$ is*

$$EPU_R(S; \theta) = \sum_{x \in S} P_R(S \rightsquigarrow x; \theta) \, EU^*(\theta | S \rightsquigarrow x) \tag{1}$$

$EVOI(S; \theta)$ is then $EPU(S; \theta) - EU^*(\theta)$, the expected improvement in decision quality given $S$. An optimal query (of fixed size $k$) is any $S$ with maximal $EVOI$, or equivalently, maximal $EPU$.

In many settings, we may wish to present a set of options to a user with the dual goals of offering a good set of recommendations and eliciting valuable information about user utility. For instance, product navigation interfaces for e-commerce sites often display a set of options from which a user can select, but also give the user a chance to critique the proposed options [24]. This provides one motivation for exploring the connection between optimal recommendation sets and optimal query sets. Moreover, even in settings where queries and recommendation are separated, we will see that query optimization can be made more efficient by exploiting this relationship.

## 3 Optimal Recommendation Sets

We consider first the problem of computing optimal recommendation sets given the system's uncertainty about the user's true utility function $w$. Given belief state $\theta$, if a single recommendation is to be made, then we should recommend the option $x^*(\theta)$ that maximizes expected utility $EU(x, \theta)$. However, there is often value in suggesting a "shortlist" containing multiple options and allowing the user to select her most preferred option. Intuitively, such a set should offer options that are *diverse* in the following sense: recommended options should be highly preferred relative to a wide range of "likely" user utility functions (relative to $\theta$) [23, 20, 4]. This stands in contrast to some recommender systems that define diversity relative to product attributes [21], with no direct reference to beliefs about user utility. It is not hard to see that "top $k$" systems, those that present the $k$ options with highest expected utility, do not generally result in good recommendation sets [20].

In broad terms, we assume that the utility of a recommendation set $S$ is the utility of its most preferred item. However, it is unrealistic to assume that users will select their most preferred item with complete accuracy [17, 15]. So as with choice queries, we assume a response model $R$ dictating the probability $P_R(S \rightsquigarrow x; \theta)$ of any choice $x$ from $S$:

**Definition 2** *The* expected utility of selection (EUS) *of recommendation set $S$ given $\theta$ and $R$ is:*

$$EUS_R(S; \theta) = \sum_{x \in S} P_R(S \rightsquigarrow x; \theta) EU(x; \theta | S \rightsquigarrow x) \tag{2}$$

We can expand the definition to rewrite $EUS_R(S; \theta)$ as:

$$EUS_R(S; \theta) = \int_W [\sum_{x \in S} P_R(S \rightsquigarrow x; w) \, u(x; w)]P(w; \theta)dw \tag{3}$$

User behavior is largely dictated by the response model $R$. In the ideal setting, users would always select the option with highest utility w.r.t. her true utility function $w$. This *noiseless model* is assumed in [20] for example. However, this is unrealistic in general. Noisy response models admit user "mistakes" and the choice of optimal sets should reflect this possibility (just as belief update does,

see Defn. 1). Possible constraints on response models include: (i) *preference bias*: a more preferred outcome in the slate given $w$ is selected with probability greater than a less preferred outcome; and (ii) *Luce's choice axiom* [17], a form of independence of irrelevant alternatives that requires that the relative probability (if not 0 or 1) of selecting any two items $x$ and $y$ from $S$ is not affected by the addition or deletion of other items from the set. We consider three different response models:

- In the *noiseless response model*, $R_{NL}$, we have $P_{NL}(S \rightsquigarrow x; w) = \prod_{y \in S} I[u(x; w) \geq u(y; w)]$ (with indicator function $I$). Then EUS becomes

$$EUS_{NL}(S; \theta) = \int_W [\max_{x \in S} u(x; w)] P(w; \theta) dw.$$

  This is identical to the *expected max* criterion of [20]. Under $R_{NL}$ we have $S \rightsquigarrow x$ iff $S \triangleright x$.

- The *constant noise model* $R_C$ assumes a multinomial distribution over choices or responses where each option $x$, apart from the most preferred option $x_w^*$ relative to $w$, is selected with (small) constant probability $P_C(S \rightsquigarrow x; w) = \beta$, with $\beta$ independent of $w$. We assume $\beta < \frac{1}{k}$, so the most preferred option is selected with probability $P_C(S \rightsquigarrow x_w^*; w) = \alpha = 1 - (k-1)\beta > \beta$. This generalizes the model used in [10, 2] to sets of any size. If $x_w^*(S)$ the optimal element in $S$ given $w$, and $u_w^*(S)$ is its utility, then EUS is:

$$EUS_C(S; \theta) = \int_W [\alpha u_w^*(S) + \sum_{y \in S - \{x_w^*(S)\}} \beta u(x; w)] P(w; \theta) dw$$

- The *logistic* response model $R_L$ is commonly used in choice modeling, and is variously known as the Luce-Sheppard [16], Bradley-Terry [11], or *mixed multinomial logit* model. Selection probabilities are given by $P_L(S \rightsquigarrow x; w) = \frac{\exp(\gamma u(x; w))}{\sum_{y \in S} \exp(\gamma u(y; w))}$, where $\gamma$ is a temperature parameter. For comparison queries (i.e., $|S| = 2$), $R_L$ is the logistic function of the difference in utility between the two options.

We now consider properties of the expected utility of selection EUS under these various models. All three models satisfy preference bias, but only $R_{NL}$ and $R_L$ satisfy Luce's choice axiom. EUS is monotone under the noiseless response model $R_{NL}$: the addition of options to a recommendation set $S$ cannot decrease its expected utility $EUS_{NL}(S; \theta)$. Moreover, say that option $x_i$ *dominates* $x_j$ relative to belief state $\theta$, if $u(x_i; w) > u(x_j; w)$ for all $w$ with nonzero density. Adding a set-wise dominated option $x$ to $S$ (i.e., an $x$ dominated by some element of $S$) does not change expected utility under $R_{NL}$: $EUS_{NL}(S \cup \{x\}; \theta) = EUS_{NL}(S; \theta)$. This stands in contrast to noisy response models, where adding dominated options might actually *decrease* expected utility.

Importantly, EUS is *submodular* for both the noiseless and the constant response models $R_C$:

**Theorem 1** *For $R \in \{R_{NL}, R_C\}$, $EUS_R$ is a submodular function of the set $S$. That is, given recommendation sets $S \subseteq Q$, option $x \notin S$, $S' = S \cup \{x\}$, and $Q' = Q \cup \{x\}$, we have:*

$$EUS_R(S'; \theta) - EUS_R(S; \theta) \geq EUS_R(Q'; \theta) - EUS_R(Q; \theta) \tag{4}$$

The proof is omitted, but simply shows that EUS has the required property of diminishing returns. Submodularity serves as the basis for a greedy optimization algorithm (see Section 5 and worst-case results on query optimization below). EUS under the commonly used logistic response model $R_L$ is not submodular, but can be related to EUS under the noiseless model—as we discuss next—allowing us to exploit submodularity of the noiseless model when optimizing w.r.t. $R_L$.

## 4 The Connection between EUS and EPU

We now develop the connection between optimal recommendation sets (using EUS) and optimal choice queries (using EPU/EVOI). As discussed above, we're often interested in sets that can serve as both good recommendations and good queries; and since EPU/EVOI can be computationally difficult, good methods for EUS-optimization can serve to generate good queries as well if we have a tight relationship between the two.

In the following, we make use of a transformation $T_{\theta,R}$ that modifies a set $S$ in such a way that EUS usually increases (and in the case of $R_{NL}$ and $R_C$ cannot decrease). This transformation is used in two ways: (i) to prove the optimality (near-optimality in the case of $R_L$) of EUS-optimal recommendation sets when used as query sets; (ii) and directly as a computationally viable heuristic strategy for generating query sets.

**Definition 3** *Let $S = \{x_1, \cdots, x_k\}$ be a set of options. Define:*
$$T_{\theta,R}(S) = \{x^*(\theta|S \rightsquigarrow x_1; R), \cdots, x^*(\theta|S \rightsquigarrow x_k; R)\}$$
*where $x^*(\theta|S \rightsquigarrow x_i; R)$ is the optimal option (in expectation) when $\theta$ is conditioned on $S \rightsquigarrow x_i$ w.r.t. $R$.*

Intuitively, $T$ (we drop the subscript when $\theta, R$ are clear from context) refines a recommendation set $S$ of size $k$ by producing $k$ updated beliefs for each possible user choice, and replacing each option in $S$ with the optimal option under the corresponding update. Note that $T$ generally produces different sets under different response models. Indeed, one could use $T$ to construct a set using one response model, and measure EUS or EPU of the resulting set under a different response model. Some of our theoretical results use this type of "cross-evaluation."

We first show that optimal recommendation sets under both $R_{NL}$ and $R_C$ are optimal (i.e., EPU/EVOI-maximizing) query sets.

**Lemma 1** $EUS_R(T_{\theta,R}(S); \theta) \geq EPU_R(S; \theta)$ *for* $R \in \{NL, C\}$

**Proof**: For the $R_{NL}$, the argument relies on partitioning $W$ w.r.t. options in $S$:

$$EPU_{NL}(S; \theta) = \sum_{i,j} P(S \triangleright x_i, T(S) \triangleright x'_j; \theta) EU(x'_i, \theta[S \triangleright x_i, T(S) \triangleright x'_j]) \tag{5}$$

$$EUS_{NL}(T(S); \theta) = \sum_{i,j} P(S \triangleright x_i, T(S) \triangleright x'_j; \theta) EU(x'_j; \theta[S \triangleright x_i, T(S) \triangleright x'_j]) \tag{6}$$

Compare the two expression componentwise: 1) If $i = j$ then the components of each expression are the same. 2) If $i \neq j$, for any $w$ with nonzero density in $\theta[S \triangleright x_i, T(S) \triangleright x'_j]$, we have $u(x'_j; w) \geq u(x'_i, w)$, thus $EU(x'_j) \geq EU(x_i)$ in the region $S \triangleright x_i, T(S) \triangleright x'_j$. Since $EUS_{NL}(T(S); \cdot) \geq EPU_{NL}(S; \cdot)$ in each component, the result follows. For $R_C$ the proof uses the same argument, along with the observation that: $EUS_C(S; \theta) = \sum_i P(S \triangleright x_i; \theta)(\alpha \, EU(x_i, \theta[S \triangleright x_i]) + \beta \sum_{j \neq i} EU(s_j, \theta[S \triangleright x_i]))$. ∎

From Lemma 1 and the fact that $EUS_R(S; \theta) \leq EPU_R(S, \theta)$, it follows that $EUS_R(T(S); \theta) \geq EUS_R(S; \theta)$. We now state the main theorem (we assume the size $k$ of $S$ is fixed):

**Theorem 2** *Assume response model $R \in \{NL, C\}$ and let $S^*$ be an optimal recommendation set. Then $S^*$ is an optimal query set: $EPU(S^*; \theta) \geq EPU(S; \theta), \forall S \in \mathbf{X}^k$*

**Proof**: Suppose $S^*$ is not an optimal query set, i.e., there is some $S$ s.t. $EPU(S; \theta) > EPU(S^*; \theta)$. Applying $T$ to $S$ gives a new query set $T(S)$, which by the results above shows: $EUS(T(S); \theta) \geq EPU(S; \theta) > EPU(S^*; \theta) \geq EUS(S^*; \theta)$. This contradicts the EUS-optimality of $S^*$. ∎

Another consequence of Lemma 1 is that posing a query $S$ involving an infeasible option is pointless: there is always a set with only elements in $\mathbf{X}$ with EPU/EVOI at least as great. This is proved by observing the lemma still holds if $T$ is redefined to allow sets containing infeasible options.

It is not hard to see that admitting noisy responses under the logistic response model $R_L$ can decrease the value of a recommendation set, i.e., $EUS_L(S; \theta) \leq EUS_{NL}(S; \theta)$. However, the loss in EUS under $R_L$ can in fact be bounded. The logistic response model is such that, if the probability of incorrect selection of some option is high, then the utility of that option must be close to that of the best item, so the relative loss in utility is small. Conversely, if the loss associated with some incorrect selection is great, its utility must be significantly less than that of the best option, rendering such an event extremely unlikely. This allows us to bound the difference between $EUS_{NL}$ and $EUS_L$ at some value $\Delta_{\max}$ that depends only on the set cardinality $k$ and on the temperature parameter $\gamma$ (we derive an expression for $\Delta_{\max}$ below):

**Theorem 3** $EUS_L(S; \theta) \geq EUS_{NL}(S; \theta) - \Delta_{\max}$.

Under $R_L$, our transformation $T_L$ does not, in general, improve the value $EUS_L(S)$ of a recommendation set $S$. However the set $T_L(S)$ is such that its value $EUS_{NL}$, *assuming selection under the noiseless model*, is greater than the expected posterior utility $EPU_L(S)$ under $R_L$:

**Lemma 2** $EUS_{NL}(T_L(S); \theta) \geq EPU_L(S; \theta)$

We use this fact below to prove the optimal recommendation set under $R_L$ is a near-optimal query under $R_L$. It has two other consequences: First, from Thm. 3 it follows that $EUS_L(T_L(S); \theta) \geq EPU_L(S; \theta) - \Delta_{\max}$. Second, EPU of the optimal query under the noiseless model is at least as great that of the optimal query under the logistic model: $EPU^*_{NL}(\theta) \geq EPU^*_L(\theta)$.[1] We now derive our main result for logistic responses: the EUS of the optimal recommendation set (and hence its EPU) is at most $\Delta_{\max}$ less than the EPU of the optimal query set.

**Theorem 4** $EUS^*_L(\theta) \geq EPU^*_L(\theta) - \Delta_{\max}$.

**Proof**: Consider the optimal query $S^*_L$ and the set $S' = T_L(S^*_L)$ obtained by applying $T_L$. From Lemma 2, $EUS_{NL}(S'; \theta) \geq EPU_L(S^*_L, \theta) = EPU^*_L(\theta)$. From Thm. 3, $EUS_L(S'; \theta) \geq EUS_{NL}(S'; \theta) - \Delta_{\max}$; and from Thm. 2, $EUS^*_{NL}(\theta) = EPU^*_{NL}(\theta)$. Thus $EUS^*_L(\theta) \geq EUS_L(S'; \theta) \geq EUS_{NL}(S'; \theta) - \Delta_{\max} \geq EPU^*_L(\theta) - \Delta_{\max}$ ∎

The loss $\Delta(S; \theta) = EUS_{NL}(S; \theta) - EUS_L(S; \theta)$ in the EUS of set $S$ due to logistic noise can be characterized as a function of the utility difference $z = u(x_1) - u(x_2)$ between options $x_1$ and $x_2$ of $S$, and integrating over the possible values of $z$ (weighted by $\theta$). For a specific value of $z \geq 0$, EUS-loss is exactly the utility difference $z$ times the probability of choosing the less preferred option under $R_L$: $1 - L(\gamma z) = L(-\gamma z)$ where $L$ is the logistic function. We have $\Delta(S; \theta) = \int_{-\infty}^{+\infty} |z| \cdot \frac{1}{1 + e^{\gamma|z|}} P(z; \theta) dz$. We derive a problem-independent upper bound on $\Delta(S; \theta)$ for any $S, \theta$ by maximizing $f(z) = z \cdot \frac{1}{1 + e^{\gamma z}}$ with $z \geq 0$. The maximal loss $\Delta_{\max} = f(z_{\max})$ for a set of two hypothetical items $s_1$ and $s_2$ is attained by having the same utility difference $u(s_1, w) - u(s_2, w) = z_{\max}$ for any $w \in W$. By imposing $\frac{\partial f}{\partial z} = 0$, we obtain $e^{-\gamma z} - \gamma z + 1 = 0$. Numerically, this yields $z_{\max} \sim 1.279\frac{1}{\gamma}$ and $\Delta_{\max} \sim 0.2785\frac{1}{\gamma}$. This bound can be expressed on a scale that is independent of the temperature parameter $\gamma$; intuitively, $\Delta_{\max}$ corresponds to a utility difference so slight that the user identifies the best item only with probability $0.56$ under $R_L$ with temperature $\gamma$. In other words, the maximum loss is so small that the user is unable able to identify the preferred item 44% of the time when asked to compare the two items in $S$. This derivation can be generalized to sets of any size $k$, yielding $\Delta^k_{\max} = \frac{1}{\gamma} \cdot \mathcal{L}_W(\frac{k-1}{e})$, where $\mathcal{L}_W(\cdot)$ is the Lambert W function.[2]

## 5  Set Optimization Strategies

We discuss several strategies for the optimization of query/recommendation sets in this section, and summarize their theoretical and computational properties. In what follows, $n$ is the number of options $|\mathbf{X}|$, $k$ the size of the query/recommendation set, and $l$ is the "cost" of Bayesian inference (e.g., the number of particles in a Monte Carlo sampling procedure).

**Exact Methods**  The naive maximization of EPU is more computationally intensive than EUS-optimization, and is generally impractical. Given a set $S$ of $k$ elements, computing $EPU(S, \theta)$ requires Bayesian update of $\theta$ for each possible response, and expected utility optimization for each such posterior. Query optimization requires this be computed for $n^k$ possible query sets. Thus EPU maximization is $O(n^{k+1}kl)$. Exact EUS optimization, while still quite demanding, is only $O(n^k kl)$ as it does not require EU-maximization in updated distributions. Thm. 2 allows us to compute optimal query sets using EUS-maximization under $R_C$ and $R_{NL}$, reducing complexity by a factor of $n$. Under $R_L$, Thm. 4 allows us to use EUS-optimization to approximate the optimal query, with a quality guarantee of $EPU^* - \Delta_{\max}$.

**Greedy Optimization**  A simple greedy algorithm can be used construct a recommendation set of size $k$ by iteratively adding the option offering the greatest improvement in value: $\arg\max_x EUS_R(S \cup \{x\}; \theta)$. Under $R_{NL}$ and $R_C$, since EUS is submodular (Thm. 1), the greedy algorithm determines a set with EUS that is within $\eta = 1 - (\frac{k-1}{k})^k$ of the optimal value

$EUS^* = EPU^*$ [9].[3] Thm. 2 again allows us to use greedy maximization of $EUS$ to determine a *query set* with similar gaurantees.

Under $R_L$, $EUS_L$ is no longer submodular. However, Lemma 2 and Thm. 3 allow us to use $EUS_{NL}$, which is submodular, as a proxy. Let $S_g$ the set determined by greedy optimization of $EUS_{NL}$. By submodularity, $\eta \cdot EUS^*_{NL} \leq EUS_{NL}(S_g) \leq EUS^*_{NL}$; we also have $EUS^*_L \leq EUS^*_{NL}$. Applying Thm. 3 to $S_g$ gives: $EUS_L(S_g) \geq EUS_{NL}(S_g) - \Delta$. Thus, we derive

$$\frac{EUS_L(S_g)}{EUS^*_L} \geq \frac{\eta \cdot EUS^*_{NL} - \Delta}{EUS^*_L} \geq \frac{\eta \cdot EUS^*_{NL} - \Delta}{EUS^*_{NL}} \geq \eta - \frac{\Delta}{EUS^*_{NL}} \qquad (7)$$

Similarly, we derive a worst-case bound for $EPU$ w.r.t. greedy EUS-optimization (using the fact that EUS is a lower bound for EPU, Thm. 3 and Thm. 2):

$$\frac{EPU_L(S_g)}{EPU^*_L} \geq \frac{EUS_L(S_g)}{EPU^*_L} \geq \frac{\eta \cdot EUS^*_{NL} - \Delta}{EPU^*_{NL}} = \frac{\eta \cdot EUS^*_{NL} - \Delta}{EUS^*_{NL}} \geq \eta - \frac{\Delta}{EUS^*_{NL}} \qquad (8)$$

Greedy maximization of $S$ w.r.t. EUS is extremely fast, $O(k^2 ln)$, or linear in the number of options $n$: it requires $O(kn)$ evaluations of $EUS$, each with cost $kl$.[4]

**Query Iteration**   The $T$ transformation (Defn. 3) gives rise to a natural heuristic method for computing, good query/recommendation sets. *Query iteration (QI)* starts with an initial set $S$, and locally optimizes $S$ by repeatedly applying operator $T(S)$ until $EUS(T(S); \theta)=EUS(S; \theta)$. QI is sensitive to the initial set $S$, which can lead to different fixed points. We consider several initialization strategies: *random* (randomly choose $k$ options), *sampling* (include $x^*(\theta)$, and sample $k - 1$ points $w^i$ from $P(w; \theta)$, and for each of these add the optimal item to $S$, while forcing distinctness) and *greedy* (initialize with the greedy set $S_g$).

We can bound the performance of QI relative to optimal query/recommendation sets assuming $R_{NL}$ or $R_C$. If QI is initialized with $S_g$, performance is no worse than greedy optimization. If initialized with an arbitrary set, we note that, because of submodularity, $EU^* \leq EUS^* \leq kEU^*$. The condition $T(S) = S$ implies $EUS(S) = EPU(S)$. Also note that, for any set $Q$, $EPU(Q) \geq EU^*$. Thus, $EUS(S) \geq \frac{1}{k} EUS^*$. This means for comparison queries ($|S| = 2$), QI achieves at least 50% of the optimal recommendation set value. This bound is tight and corresponds to the *singleton degenerate set* $S_d = \{x^*(\theta), .., x^*(\theta)\} = \{x^*(\theta)\}$. This solution is problematic since $T(S_d) = S_d$ and has EVOI of zero. However, under $R_{NL}$, QI with sampling initialization avoids this fixed point provably by construction, always leading to a query set with positive EVOI.

Complexity of one iteration of QI is $O(nk + lk)$, i.e., linear in the number of options, exactly like Greedy. However, in practice it is much faster than Greedy since typically $k << l$. While we have no theoretical results that limit the iterations required by QI to converge, in practice, a fixed point is reached in very quickly (see below).

**Evaluation**   We compare the strategies above empirically on choice problems with random user utility functions using both noiseless and noisy response models.[5]

Bayesian inference is realized by a Monte Carlo method with importance sampling (particle weights are determined by applying the response model to observed responses). To overcome the problem of particle degeneration (most particles eventually have low or zero weight), we use slice-sampling [18] to regenerate particles w.r.t. to the response-updated belief state $\theta$ whenever the *effective number of samples* drops significantly (50000 particles were used in the simulations). Figure 1(a) shows the average loss of our strategies in an apartment rental dataset, with 187 outcomes, each characterized by 10 attributes (either numeric or categorical with domain sizes 2–6), when asking pairwise comparison queries with noiseless responses. We note that greedy performs almost as well as exact optimization, and the optimal item is found in roughly 10–15 queries. Query iteration performs reasonably well when initialized with sampling, but poorly with random seeds.

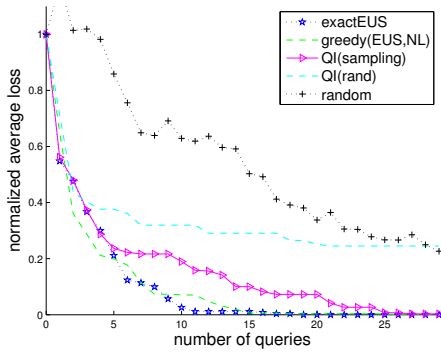
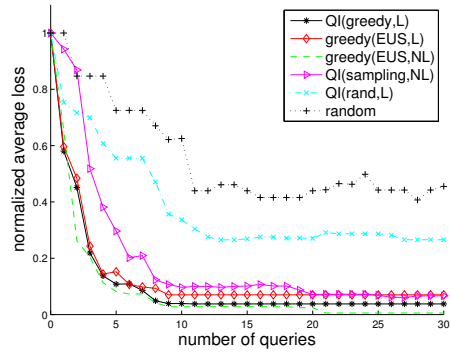

(a) Average Loss (187 outcomes, 30 runs, $R_{NL}$)      (b) Average Loss (506 outcomes, 30 runs, $R_L$)

In the second experiment, we consider the *Boston Housing* dataset with 506 items (1 binary and 13 continous attributes) and a logistic noise model for responses with $\gamma = 1$. We compare the greedy and QI strategies (exact methods are impractical on problems of this size) in Figure 1(b); we also consider a hybrid *greedy(EUS,NL)* strategy that optimizes "assuming" noiseless responses, but is evaluated using the true response model $R_L$. *QI(sampling)* is more efficient when using $T_{NL}$ instead of $T_L$ and this is the version plotted. Overall these experiments show that (greedy or exact) maximization of EUS is able to find optimal—or near-optimal when responses are noisy—query sets. Finally, we compare query optimization times on the two datasets in the following table:

|  | exactEPU | exactEUS | greedy(EPU,L) | QI(greedy(EUS,L)) | greedy(EUS,L) | greedy(EUS,NL) | QI(sampling) | QI(rand) |
|---|---|---|---|---|---|---|---|---|
| n=30, k=2 | **47.3s** | 10.3s | 1.5s | 0.76s | 0.65s | 0.12s | 0.11s | **0.11s** |
| n=187, k=2 | **1815s** | 405s | 9.19s | 2.07s | 1.97s | 1.02s | **0.15s** | 0.17s |
| n=187, k=4 | - | 10000s | 39.7s | 7.89s | 7.71s | 1.86s | **0.16s** | 0.19s |
| n=187, k=6 | - | - | 87.1s | 15.7s | 15.4s | 2.55s | **0.51s** | 0.64s |
| n=506, k=2 | - | - | 14.6s | 4.09s | 3.99s | 0.93s | **0.05s** | 0.06s |
| n=506, k=4 | - | - | 64.9s | 15.4s | 15.2s | 1.12s | **0.08s** | 0.10s |
| n=506, k=6 | - | - | 142s | 32.9s | 32.8s | 1.53s | **0.09s** | 0.13s |

Among our strategies, QI is certainly most efficient computationally, and is best suited to large outcome spaces. Interestingly, QI is often faster with sampling initialization than with random initialization because it needs fewer iteration on average before convergence (3.1 v.s. 4.0).

# 6 Conclusions

We have provided a novel analysis of set-based recommendations in Bayesian recommender systems, and have shown how it is offers a tractable means of generating myopically optimal or near-optimal choice queries for preference elicitation. We examined several user response models, showing that optimal recommendation sets are EVOI-optimal queries under noiseless and constant noise models; and that they are near-optimal under the logistic/Luce-Sheppard model (both theoretically and practically). We stress that our results are general and do not depend on the specific implementation of Bayesian update, nor on the specific form of the utility function. Our greedy strategies—exploiting submodularity of EUS computation—perform very well in practice and have theoretical approximation guarantees. Finally our experimental results demonstrate that query iteration, a simple local search strategy, is especially well-situated to large decision spaces.

A number of important directions for future research remain. Further theoretical and practical investigation of local search strategies such as query iteration is important. Another direction is the development of strategies for Bayesian recommendation and elicitation in large-scale configuration problems, e.g., where outcomes are specified by a CSP, and for sequential decision problems (such as MDPs with uncertain rewards). Finally, we are interested in elicitation strategies that combine probabilistic and regret-based models.

**Acknowledgements**   The authors would like to thank Iain Murray and Cristina Manfredotti for helpful discussion on Monte Carlo methods, sampling techniques and particle filters. This research was supported by NSERC.

## Footnotes

[1] $EPU_L(S; \theta)$ is *not* necessarily less than $EPU_{NL}(S; \theta)$: there are sets $S$ for which a noisy response might be "more informative" than a noiseless one. However, this is not the case for optimal query sets.

[2] Lambert W, or *product-log*, is defined as the principal value of the inverse of $x \cdot e^x$. The loss-maximizing set $S_{\max}$ may contain infeasible outcomes; so in practice loss may be much lower.

[3]This is 75% for comparison queries ($k = 2$) and at worst 63% (as $k \to \infty$).

[4]A practical speedup can be achieved by maintaining a priority queue of outcomes sorted by their potential EUS-contribution (monotonically decreasing due to submodularity). When choosing the item to add to the set, we only need to evaluate a few outcomes at the top of the queue (*lazy evaluation*).

[5]Utility priors are mixtures of 3 Gaussians with $\mu = U[0, 10]$ and $\sigma = \mu/3$ for each component.

# References

[1] Ali Abbas. Entropy methods for adaptive utility elicitation. *IEEE Transactions on Systems, Science and Cybernetics*, 34(2):169–178, 2004.

[2] Craig Boutilier. A POMDP formulation of preference elicitation problems. In *Proceedings of the Eighteenth National Conference on Artificial Intelligence (AAAI-02)*, pp.239–246, Edmonton, 2002.

[3] Craig Boutilier, Relu Patrascu, Pascal Poupart, and Dale Schuurmans. Constraint-based optimization and utility elicitation using the minimax decision criterion. *Artifical Intelligence*, 170(8–9):686–713, 2006.

[4] Craig Boutilier, Richard S. Zemel, and Benjamin Marlin. Active collaborative filtering. In *Proc. 19th Conference on Uncertainty in Artificial Intelligence (UAI-03)*, pp.98–106, Acapulco, 2003.

[5] Darius Braziunas and Craig Boutilier. Minimax regret-based elicitation of generalized additive utilities. In *Proc. 23rd Conference on Uncertainty in Artificial Intelligence (UAI-07)*, pp.25–32, Vancouver, 2007.

[6] U. Chajewska and D. Koller. Utilities as random variables: Density estimation and structure discovery. In *Proc. 16th Conference on Uncertainty in Artificial Intelligence (UAI-00)*, pp.63–71, Stanford, 2000.

[7] U. Chajewska, D. Koller, and R. Parr. Making rational decisions using adaptive utility elicitation. In *Proc. 17th National Conference on Artificial Intelligence (AAAI-00)*, pp.363–369, Austin, TX, 2000.

[8] Peter C. Fishburn. Interdependence and additivity in multivariate, unidimensional expected utility theory. *International Economic Review*, 8:335–342, 1967.

[9] L. A. Wolsey G. L. Nemhauser and M. L. Fisher. An analysis of approximations for maximizing submodular set functions. *Mathematical Programming*, 14(1):265–294, December 1978.

[10] Krzysztof Gajos and Daniel S. Weld. Preference elicitation for interface optimization. In Patrick Baudisch, Mary Czerwinski, and Dan R. Olsen, editors, *UIST*, pp.173–182. ACM, 2005.

[11] Shengbo Guo and Scott Sanner. Real-time multiattribute bayesian preference elicitation with pairwise comparison queries. In *Proceedings of the 13th International Conference on Artificial Intelligence and Statistics (AISTATS-10)*, Sardinia, Italy, 2010.

[12] V. S. Iyengar, J. Lee, and M. Campbell. Q-Eval: Evaluating multiple attribute items using queries. In *Proceedings of the Third ACM Conference on Electronic Commerce*, pp.144–153, Tampa, FL, 2001.

[13] Ralph L. Keeney and Howard Raiffa. *Decisions with Multiple Objectives: Preferences and Value Tradeoffs*. Wiley, New York, 1976.

[14] J. A. Konstan, B. N. Miller, D. Maltz, J. L. Herlocker, L. R. Gordon, and J. Riedl. Grouplens: Applying collaborative filtering to usenet news. *Communications of the ACM*, 40(3):77–87, 1997.

[15] Jordan J. Louviere, David A. Hensher, and Joffre D. Swait. *Stated Choice Methods: Analysis and Application*. Cambridge University Press, Cambridge, 2000.

[16] Christopher G. Lucas, Thomas L. Griffiths, Fei Xu, and Christine Fawcett. A rational model of preference learning and choice prediction by children. In *Proceedings of the Twenty-Second Annual Conference on Neural Information Processing Systems, Vancouver, Canada, 2008*, pp.985–992, 2008.

[17] Robert D. Luce. *Individual choice behavior: a theoretical analysis*. Wiley, New York, 1959.

[18] Radford M. Neal. Slice sampling. *The Annals of Statistics*, 31(3):705–70, 2003.

[19] A. Ng and S. Russell. Algorithms for inverse reinforcement learning. In *Proc. 17th International Conference on Machine Learning (ICML-00)*, pp.663–670, Stanford, CA, 2000.

[20] Robert Price and Paul R. Messinger. Optimal recommendation sets: Covering uncertainty over user preferences. In *Proceedings of the Twentieth National Conference on Artificial Intelligence (AAAI'05)*, pp.541–548, 2005.

[21] James Reilly, Kevin McCarthy, Lorraine McGinty, and Barry Smyth. Incremental critiquing. *Knowledge-Based Systems*, 18(4–5):143–151, 2005.

[22] Olivier Toubia, John Hauser, and Duncan Simester. Polyhedral methods for adaptive choice-based conjoint analysis. *Journal of Marketing Research*, 41:116–131, 2004.

[23] Paolo Viappiani and Craig Boutilier. Regret-based optimal recommendation sets in conversational recommender systems. In *Proceedings of the 3rd ACM Conference on Recommender Systems (RecSys09)*, pp.101–108, New York, 2009.

[24] Paolo Viappiani, Boi Faltings, and Pearl Pu. Preference-based search using example-critiquing with suggestions. *Journal of Artificial Intelligence Research*, 27:465–503, 2006.

